# Strategies for Teaching Layered Networks Classification Tasks

Ben S. Wittner [1] and John S. Denker
AT&T Bell Laboratories
Holmdel, New Jersey 07733

## Abstract

There is a widespread misconception that the delta-rule is in some sense guaranteed to work on networks without hidden units. As previous authors have mentioned, there is no such guarantee for classification tasks. We will begin by presenting explicit counter-examples illustrating two different interesting ways in which the delta rule can fail. We go on to provide conditions which do guarantee that gradient descent will successfully train networks without hidden units to perform two-category classification tasks. We discuss the generalization of our ideas to networks with hidden units and to multi-category classification tasks.

## The Classification Task

Consider networks of the form indicated in figure 1. We discuss various methods for training such a network, that is for adjusting its weight vector, $\mathbf{w}$. If we call the input $\mathbf{v}$, the output is $g(\mathbf{w} \cdot \mathbf{v})$, where $g$ is some function.

The classification task we wish to train the network to perform is the following. Given two finite sets of vectors, $F_1$ and $F_2$, output a number greater than zero when a vector in $F_1$ is input, and output a number less than zero when a vector in $F_2$ is input. Without significant loss of generality, we assume that $g$ is odd (i.e. $g(-s) = -g(s)$). In that case, the task can be reformulated as follows. Define [2]

$$F := F_1 \cup \{-\mathbf{v} \text{ such that } \mathbf{v} \in F_2\} \tag{1}$$

and output a number greater than zero when a vector in $F$ is input. The former formulation is more natural in some sense, but the later formulation is somewhat more convenient for analysis and is the one we use. We call vectors in $F$, *training vectors*.

## A Class of Gradient Descent Algorithms

We denote the solution set by

$$W := \{\mathbf{w} \text{ such that } g(\mathbf{w} \cdot \mathbf{v}) > 0 \text{ for all } \mathbf{v} \in F\}, \tag{2}$$

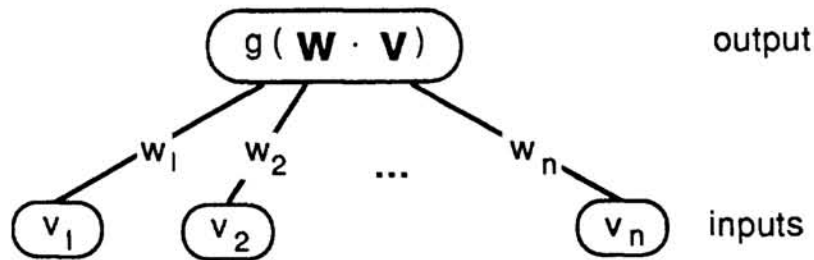

Figure 1: a simple network

and we are interested in rules for finding some weight vector in $W$. We restrict our attention to rules based upon gradient descent down error functions $E(\mathbf{w})$ of the form

$$E(\mathbf{w}) = \sum_{\mathbf{v} \in F} h(\mathbf{w} \cdot \mathbf{v}). \tag{3}$$

The delta-rule is of this form with

$$h(\mathbf{w} \cdot \mathbf{v}) = h_\delta(\mathbf{w} \cdot \mathbf{v}) := \frac{1}{2}(b - g(\mathbf{w} \cdot \mathbf{v}))^2 \tag{4}$$

for some positive number $b$ called the *target* (Rumelhart, McClelland, et al.). We call the delta rule error function $E_\delta$.

## Failure of Delta-rule Using Obtainable Targets

Let $g$ be any function that is odd and differentiable with $g'(s) > 0$ for all $s$. In this section we assume that the target $b$ is in the range of $g$. We construct a set $F$ of training vectors such that even though $W$ is not empty, there is a local minimum of $E_\delta$ not located in $W$. In order to facilitate visualization, we begin by assuming that $g$ is linear. We will then indicate why the construction works for the nonlinear case as well. We guess that this is the type of counter-example alluded to by Duda and Hart (p. 151) and by Minsky and Papert (p. 15).

The input vectors are two dimensional. The arrows in figure 2 represent the training vectors in $F$ and the shaded region is $W$. There is one training vector, $\mathbf{v}^1$, in the second quadrant, and all the rest are in the first quadrant. The training vectors in the first quadrant are arranged in pairs symmetric about the ray $R$ and ending on the line $L$. The line $L$ is perpendicular to $R$, and intersects $R$ at unit distance from the origin. Figure 2 only shows three of those symmetric pairs, but to make this construction work we might need many. The point $\mathbf{p}$ lies on $R$ at a distance of $g^{-1}(b)$ from the origin.

We first consider the contribution to $E_\delta$ due to any single training vector, $\mathbf{v}$. The contribution is

$$(1/2)(b - g(\mathbf{w} \cdot \mathbf{v}))^2, \tag{5}$$

and is represented in figure 3 in the $z$-direction. Since $g$ is linear and since $b$ is in the

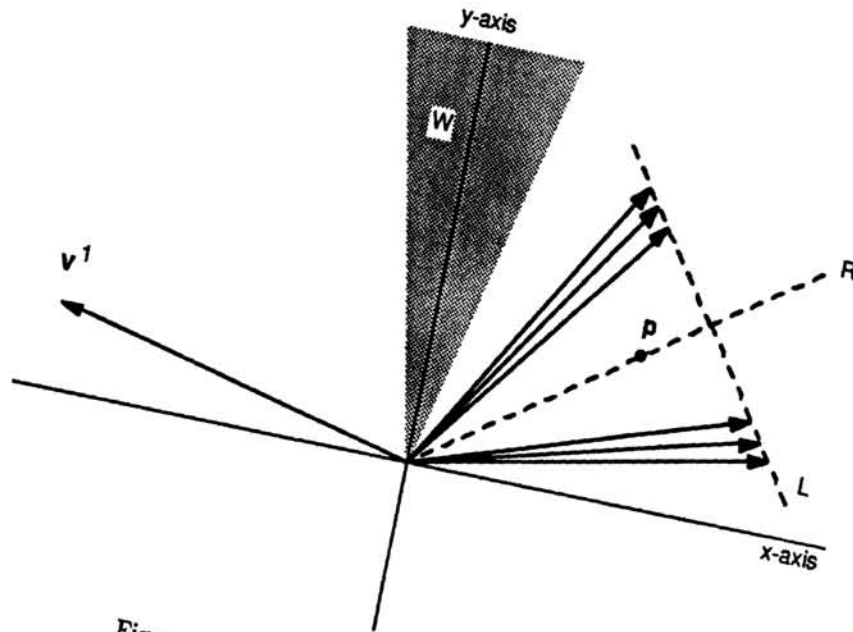

Figure 2: Counter-example for obtainable targets

range of $g$, the contribution is a quadratic trough with bottom on a line in the $xy$-plane and perpendicular to $\mathbf{v}$. If $\mathbf{v}$ is one of the training vectors in the first quadrant, then the point $\mathbf{p}$ lies along the bottom line of the trough.

Now we consider the contribution to the error function due to one of the symmetric pairs. It is the sum of two quadratic troughs with bottom lines intersecting only at the point $\mathbf{P}$. So it is a quadratic bowl with bottom point at $\mathbf{p}$.

Next we consider the contribution to $E_\delta$ due to all the training vectors in the first quadrant. It is a sum of quadratic bowls, all with bottom at $\mathbf{p}$. So it is itself a quadratic bowl with bottom at $\mathbf{p}$ and it can be made arbitrarily steep by having arbitrarily many of the symmetric pairs. Let us call this contribution $E_0$.

We denote by $E_1$ the contribution to $E_\delta$ due to $\mathbf{v}^1$. $E_1$ is a quadratic trough and $E_0$ is a quadratic bowl, so $E_\delta = E_0 + E_1$. $E_1$ is a quadratic bowl with a single minimum. That minimum is closer to $W$ than is $\mathbf{p}$, but if the bowl defined by $E_0$ is sufficiently steep, the minimum will be sufficiently near $\mathbf{p}$ so as to not be in $W$. Q.E.D.

Since $E_\delta$ is a quadratic function of $\mathbf{w}$, it is easy to compute directly the zeroes of its gradient. In this way, we have confirmed the conclusion of the conceptual argument presented above.

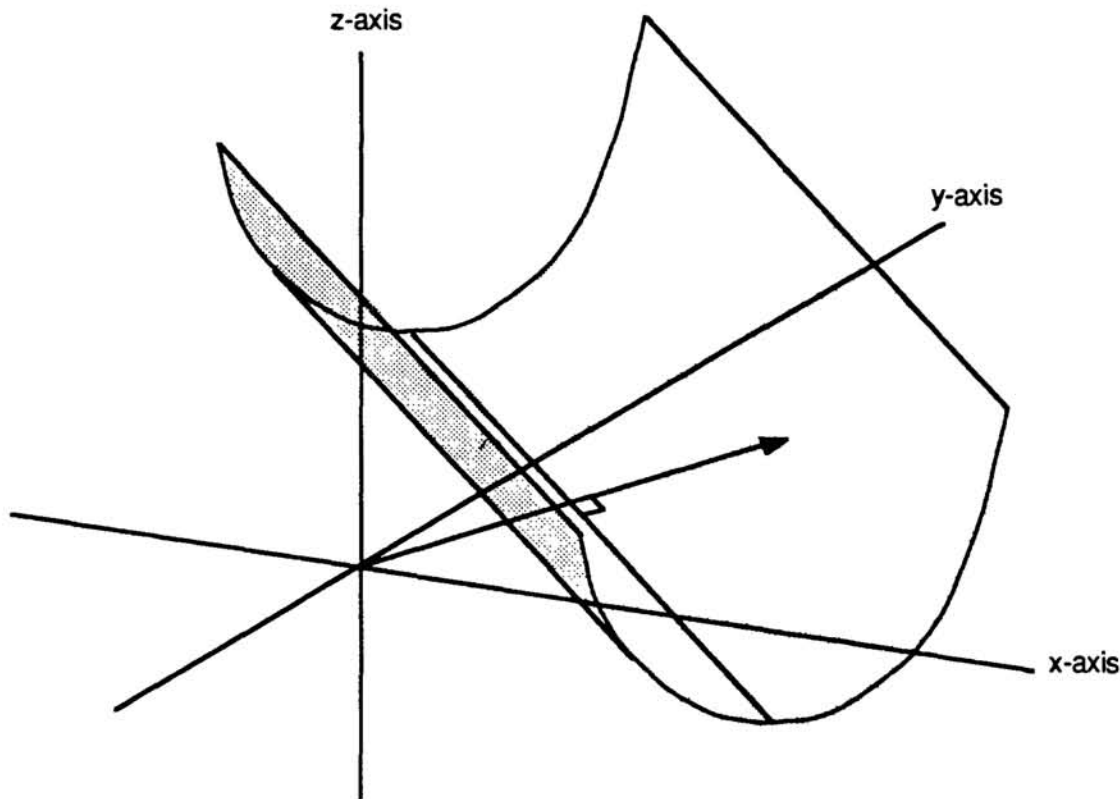

Figure 3: Error surface

We now remove the assumption that $g$ is linear. The key observation is that

$$dh_\delta/ds \equiv h_\delta'(s) = (b - g(s))(-g'(s)) \tag{6}$$

still only has a single zero at $g^{-1}(b)$ and so $h(s)$ still has a single minimum at $g^{-1}(b)$. The contribution to $E_\delta$ due to the training vectors in the first quadrant therefore still has a global minimum on the $xy$-plane at the point $\mathbf{p}$. So, as in the linear case, if there are enough symmetric pairs of training vectors in the first quadrant, the value of $E_0$ at $\mathbf{p}$ can be made arbitrarily lower than the value along some circle in the $xy$-plane centered around $\mathbf{p}$, and $E_\delta = E_0 + E_1$ will have a local minimum arbitrarily near $\mathbf{p}$. Q.E.D.

## Failure of Delta-rule Using Unobtainable Targets

We now consider the case where the target $b$ is *greater* than any number in the range of $g$. The kind of counter-example presented in the previous section no longer exists, but we will show that for some choices of $g$, including the traditional choices, the delta rule can still fail. Specifically, we construct a set $F$ of training vectors such that even though $W$ is not empty, for some choices of initial weights, the path traced out by going down the gradient of $E_\delta$ never enters $W$.

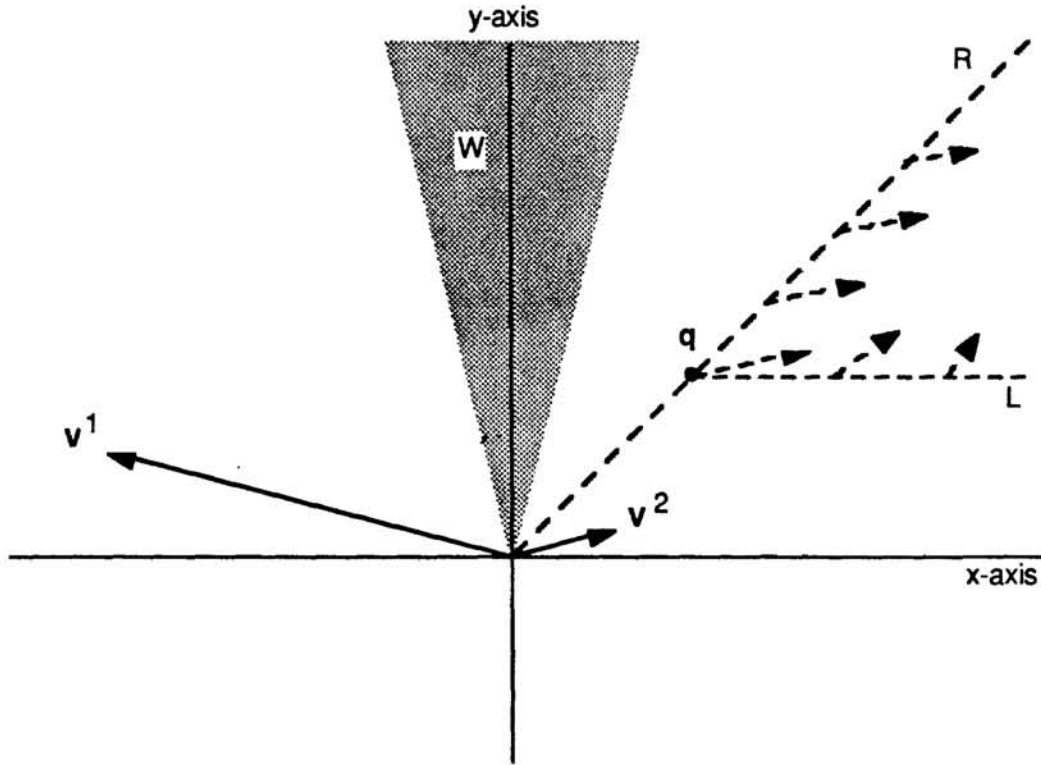

Figure 4: Counter-example for unobtainable targets

We suppose that $g$ has the following property. There exists a number $r > 0$ such that

$$\lim_{s \to \infty} \frac{h_\delta'(-rs)}{h_\delta'(s)} = 0. \tag{7}$$

An example of such a $g$ is

$$g(s) = \tanh(s) = \frac{2}{1 + e^{-2s}} - 1, \tag{8}$$

for which any $r$ greater than 1 will do.

The solid arrows in figure 4 represent the training vectors in $F$ and the more darkly shaded region is $W$. The set $F$ has two elements,

$$\mathbf{v}^1 = \begin{bmatrix} -2 \\ 1 \end{bmatrix} \qquad \text{and} \qquad \mathbf{v}^2 = \left(\frac{1}{r}\right)\left(\frac{1}{3}\right)\begin{bmatrix} 2 \\ 1 \end{bmatrix} \tag{9}$$

The dotted ray, $R$ lies on the diagonal $\{y = x\}$.

Since

$$E_\delta(\mathbf{w}) = h_\delta(\mathbf{w} \cdot \mathbf{v}^1) + h_\delta(\mathbf{w} \cdot \mathbf{v}^2), \tag{10}$$

the gradient descent algorithm follows the vector field

$$-\nabla E(\mathbf{w}) = -h_\delta'(\mathbf{w} \cdot \mathbf{v}^1)\mathbf{v}^1 - h_\delta'(\mathbf{w} \cdot \mathbf{v}^2)\mathbf{v}^2. \tag{11}$$

The reader can easily verify that for all $\mathbf{w}$ on $R$,

$$\mathbf{w} \cdot \mathbf{v}^1 = -r\mathbf{w} \cdot \mathbf{v}^2. \tag{12}$$

So by equation (7), if we constrain $\mathbf{w}$ to move along $R$,

$$\lim_{\mathbf{w} \to \infty} \frac{-h_\delta'(\mathbf{w} \cdot \mathbf{v}^1)}{-h_\delta'(\mathbf{w} \cdot \mathbf{v}^2)} = 0. \tag{13}$$

Combining equations (11) and (13) we see that there is a point $\mathbf{q}$ somewhere on $R$ such that beyond $\mathbf{q}$, $-\nabla E(\mathbf{w})$ points into the region to the right of $R$, as indicated by the dotted arrows in figure 4.

Let $L$ be the horizontal ray extending to the right from $\mathbf{q}$. Since for all $s$,

$$g'(s) > 0 \qquad \text{and} \qquad b > g(s), \tag{14}$$

we get that

$$-h_\delta'(s) = (b - g(s))g'(s) > 0. \tag{15}$$

So since both $\mathbf{v}^1$ and $\mathbf{v}^2$ have a positive $y$-component, $-\nabla E(\mathbf{w})$ also has a positive $y$-component for all $\mathbf{w}$. So once the algorithm following $-\nabla E$ enters the region above $L$ and to the right of $R$ (indicated by light shading in figure 4), it never leaves. Q.E.D.

## Properties to Guarantee Gradient Descent Learning

In this section we present three properties of an error function which guarantee that gradient descent will not fail to enter a non-empty $W$.

We call an error function of the form presented in equation (3) *well formed* if $h$ is differentiable and has the following three properties.

1. For all $s$, $-h'(s) \geq 0$ (i.e. $h$ does not push in the wrong direction).

2. There exists some $\epsilon > 0$ such that $-h'(s) \geq \epsilon$ for all $s \leq 0$ (i.e. $h$ keeps pushing if there is a misclassification).

3. $h$ is bounded below.

**Proposition 1** *If the error function is well formed, then gradient descent is guaranteed to enter $W$, provided $W$ is not empty.*

The proof proceeds by contradiction. Suppose for some starting weight vector the path traced out by gradient descent never enters $W$. Since $W$ is not empty, there is some non-zero $\mathbf{w}^*$ in $W$. Since $F$ is finite,

$$\lambda := \min\{\mathbf{w}^* \cdot \mathbf{v} \text{ such that } \mathbf{v} \in F\} > 0. \tag{16}$$

Let $\mathbf{w}(t)$ be the path traced out by the gradient descent algorithm. So

$$\mathbf{w}'(t) = -\nabla E(\mathbf{w}(t)) = \sum_{\mathbf{v} \in F} -h'(\mathbf{w}(t) \cdot \mathbf{v})\mathbf{v} \qquad \text{for all } t. \tag{17}$$

Since we are assuming that at least one training vector is misclassified at all times, by properties 1 and 2 and equation (17),

$$\mathbf{w}^* \cdot \mathbf{w}'(t) \geq \epsilon\lambda \qquad \text{for all } t. \tag{18}$$

So

$$|\mathbf{w}'(t)| \geq \epsilon\lambda/|\mathbf{w}^*| =: \xi > 0 \qquad \text{for all } t. \tag{19}$$

By equations (17) and (19),

$$dE(\mathbf{w}(t))/dt = \nabla E \cdot \mathbf{w}'(t) = -\mathbf{w}'(t) \cdot \mathbf{w}'(t) \leq -\xi^2 < 0 \qquad \text{for all } t. \tag{20}$$

This means that

$$E(\mathbf{w}(t)) \to -\infty \quad \text{as} \quad t \to \infty. \tag{21}$$

But property 3 and the fact that $F$ is finite guarantee that $E$ is bounded below. This contradicts equation (21) and finishes the proof.

## Consensus and Compromise

So far we have been concerned with the case in which $F$ is separable (i.e. $W$ is not empty). What kind of behavior do we desire in the non-separable case? One might hope that the algorithm will choose weights which produce correct results for as many of the training vectors as possible. We suggest that this is what gradient descent using a well formed error function does.

From investigations of many well formed error functions, we suspect the following well formed error function is representative. Let $g(s) = s$, and for some $b > 0$, let

$$h(s) = \begin{cases} (b-s)^2 & \text{if } s \leq b; \\ 0 & \text{otherwise.} \end{cases} \tag{22}$$

In all four frames of figure 5 there are three training vectors. Training vectors 1 and 2 are held fixed while 3 is rotated to become increasingly inconsistent with the others. In frames (i) and (ii) $F$ is separable. The training set in frame (iii) lies just on the border between separability and non-separability, and the one in frame (iv) is in the interior of

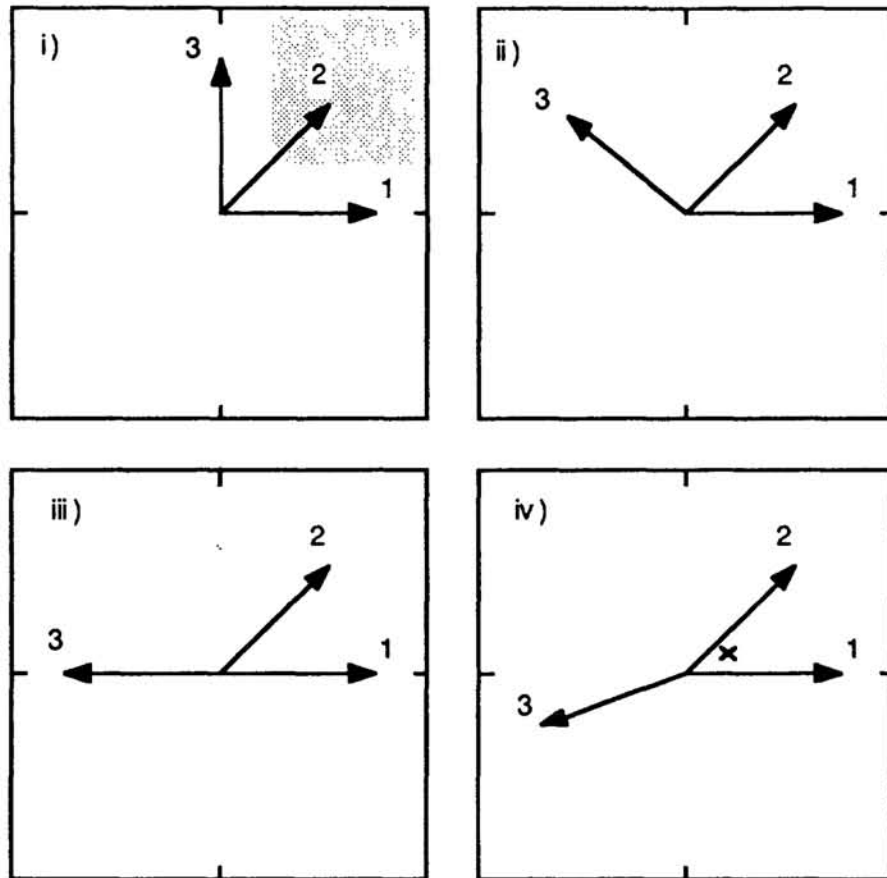

Figure 5: The transition between seperability and non-seperability

the non-separable regime. Regardless of the position of vector 3, the global minimum of the error function is the only minimum.

In frames (i) and (ii), the error function is zero on the shaded region and the shaded region is contained in $W$. As we move training vector number 3 towards its position in frame (iii), the situation remains the same except the shaded region moves arbitrarily far from the origin. At frame (iii) there is a discontinuity; the region on which the error function is at its global minimum is now the one-dimensional ray indicated by the shading. Once training vector 3 has moved into the interior of the non-separable regime, the region on which the error function has its global minimum is a point closer to training vectors 1 and 2 than to 3 (as indicated by the "x" in frame (iv)).

If all the training vectors can be satisfied, the algorithm does so; otherwise, it tries to satisfy as many as possible, and there is a discontinuity between the two regimes. We summarize this by saying that it finds a consensus if possible, otherwise it devises a compromise.

## Hidden Layers

For networks with hidden units, it is probably impossible to prove anything like proposition 1. The reason is that even though property 2 assures that the top layer of weights

gets a non-vanishing error signal for misclassified inputs, the lower layers might still get a vanishingly weak signal if the units above them are operating in the saturated regime.

We believe it is nevertheless a good idea to use a well formed error function when training such networks. Based upon a probabilistic interpretation of the output of the network, Baum and Wilczek have suggested using an entropy error function (we thank J.J. Hopfield and D.W. Tank for bringing this to our attention). Their error function is well formed. Levin, Solla, and Fleisher report simulations in which switching to the entropy error function from the delta-rule introduced an order of magnitude speed-up of learning for a network with hidden units.

## Multiple Categories

Often one wants to classify a given input vector into one of many categories. One popular way of implementing multiple categories in a feed-forward network is the following. Let the network have one output unit for each category. Denote by $o_j^v(\mathbf{w})$ the output of the $j$-th output unit when input $\mathbf{v}$ is presented to the network having weights $\mathbf{w}$. The network is considered to have classified $\mathbf{v}$ as being in the $k$-th category if

$$o_k^v(\mathbf{w}) > o_j^v(\mathbf{w}) \quad \text{for all } j \neq k. \tag{23}$$

The way such a network is usually trained is the generalized delta-rule (Rumelhart, McClelland, et al.). Specifically, denote by $c(\mathbf{v})$ the desired classification of $\mathbf{v}$ and let

$$b_j^v := \begin{cases} b & \text{if } j = c(\mathbf{v}); \\ -b & \text{otherwise,} \end{cases} \tag{24}$$

for some target $b > 0$. One then uses the error function

$$E(\mathbf{w}) := \sum_{\mathbf{v}} \sum_j \left( b_j^v - o_j^v(\mathbf{w}) \right)^2. \tag{25}$$

This formulation has several bothersome aspects. For one, the error function is not will formed. Secondly, the error function is trying to adjust the outputs, but what we really care about is the *differences* between the outputs. A symptom of this is the fact that the change made to the weights of the connections to any output unit does not depend on any of the weights of the connections to any of the other output units.

To remedy this and also the other defects of the delta rule we have been discussing, we suggest the following. For each $\mathbf{v}$ and $j$, define the relative coordinate

$$\beta_j^v(\mathbf{w}) := o_{c(\mathbf{v})}^v(\mathbf{w}) - o_j^v(\mathbf{w}). \tag{26}$$

What we really want is all the $\beta$ to be positive, so use the error function

$$E(\mathbf{w}) := \sum_{\mathbf{v}} \sum_{j \neq c(\mathbf{v})} h\left(\beta_j^{\mathbf{v}}(\mathbf{w})\right) \qquad (27)$$

for some well formed $h$. In the simulations we have run, this does not always help, but sometimes it helps quite a bit.

We have one further suggestion. Property 2 of a well formed error function (and the fact that derivatives are continuous) means that the algorithm will not be completely satisfied with positive $\beta$; it will try to make them greater than zero by some non-zero margin. That is a good thing, because the training vectors are only representatives of the vectors one wants the network to correctly classify. Margins are critically important for obtaining robust performance on input vectors not in the training set. The problem is that the margin is expressed in meaningless units; it makes no sense to use the same numerical margin for an output unit which varies a lot as is used for an output unit which varies only a little. We suggest, therefore, that for each $j$ and $\mathbf{v}$, keep a running estimate of $\sigma_j^{\mathbf{v}}(\mathbf{w})$, the variance of $\beta_j^{\mathbf{v}}(\mathbf{w})$, and replace $\beta_j^{\mathbf{v}}(\mathbf{w})$ in equation (27) by

$$\beta_j^{\mathbf{v}}(\mathbf{w})/\sigma_j^{\mathbf{v}}(\mathbf{w}). \qquad (28)$$

Of course, when beginning the gradient descent, it is difficult to have a meaningful estimate of $\sigma_j^{\mathbf{v}}(\mathbf{w})$ because $\mathbf{w}$ is changing so much, but as the algorithm begins to converge, your estimate can become increasingly meaningful.

## Footnotes

[1] Currently at NYNEX Science and Technology, 500 Westchester Ave., White Plains, NY 10604

[2] We use both $A := B$ and $B =: A$ to denote "$A$ is by definition $B$".

# References

1. David Rumelhart, James McClelland, and the PDP Research Group, *Parallel Distributed Processing*, MIT Press, 1986

2. Richard Duda and Peter Hart, *Pattern Classification and Scene Analysis*, John Wiley & Sons, 1973.

3. Marvin Minsky and Seymour Papert, "On Perceptrons", Draft, 1987.

4. Eric Baum and Frank Wilczek, these proceedings.

5. Esther Levin, Sara A. Solla, and Michael Fleisher, private communications.
